# INTERACTION AMONG OCULARITY, RETINOTOPY AND ON-CENTER/OFF-CENTER PATHWAYS DURING DEVELOPMENT

**Shigeru Tanaka**
Fundamental Research Laboratories, NEC Corporation,
34 Miyukigaoka, Tsukuba, Ibaraki 305, Japan

## ABSTRACT

The development of projections from the retinas to the cortex is mathematically analyzed according to the previously proposed thermodynamic formulation of the self-organization of neural networks. Three types of submodality included in the visual afferent pathways are assumed in two models: model (A), in which the ocularity and retinotopy are considered separately, and model (B), in which on-center/off-center pathways are considered in addition to ocularity and retinotopy. Model (A) shows striped ocular dominance spatial patterns and, in ocular dominance histograms, reveals a dip in the binocular bin. Model (B) displays spatially modulated irregular patterns and shows single-peak behavior in the histograms. When we compare the simulated results with the observed results, it is evident that the ocular dominance spatial patterns and histograms for models (A) and (B) agree very closely with those seen in monkeys and cats.

## 1  INTRODUCTION

A recent experimental study has revealed that spatial patterns of ocular dominance columns (ODCs) observed by autoradiography and profiles of the ocular dominance histogram (ODH) obtained by electrophysiological experiments differ greatly between monkeys and cats. ODCs for cats in the tangential section appear as beaded patterns with an irregularly fluctuating bandwidth (Anderson, Olavarria and Van Sluyters 1988); ODCs for monkeys are likely to be straight parallel stripes (Hubel, Wiesel and LeVay, 1977). The typical ODH for cats has a single peak in the middle of the ocular dominance corresponding to balanced response in ocularity (Wiesel and Hubel, 1974). In contrast to this, the ODH for monkeys has a dip in the middle of the ocular dominance (Hubel and Wiesel, 1963). Furthermore, neurons in the input layer of the cat's primary visual cortex exhibit orientation selectivity, while those of the monkey do not.

Through these comparisons, we can observe distinct differences in the anatomical and physiological properties of neural projections from the retinas to the visual cortex in monkeys and cats. To obtain a better understanding of these differences, theoretical analyses of interactions among ocularity, retinotopy and on-center/off-center pathways during visual

cortical development were performed with computer simulation based on the previously proposed thermodynamic formulation of the self-organization of neural networks (Tanaka, 1990).

Two models for the development of the visual afferent pathways are assumed: model (A), in which the development of ocular dominance and retinotopic order is taken into account, and model (B), in which the development of on-center/off-center pathway terminals is considered in addition to ocular dominance and retinotopic order.

## 2  MODEL DESCRIPTION

The synaptic connection density of afferent fibers from the lateral geniculate nucleus (LGN) in a local equilibrium state is represented by the Potts spin variables $\sigma_{j,k,\mu}$'s because of their strong winner-take-all process (Tanaka, 1990). The following function $\pi_{eq}(\{\sigma_{j,k,\mu}\})$ gives the distribution of the Potts spins in equilibrium:

$$\pi_{eq}(\{\sigma_{j,k,\mu}\}) = \frac{1}{Z} exp(-\frac{H(\{\sigma_{j,k,\mu}\})}{T}) \qquad (1)$$

$$\text{with } Z = \sum_{\{\sigma_{j,k,\mu}=1,0\}} exp(-\frac{H(\{\sigma_{j,k,\mu}\})}{T}) . \qquad (2)$$

The Hamiltonian $H$ in the argument of the exponential function in (1) and (2) determines the behavior of this spin system at the effective temperature $T$, where $H$ is given by

$$H = - \sum_{jj'} \sum_{\mu\mu'} \sum_{\substack{k \in B_j \\ k' \in B_{j'}}} V_{jj'}^{VC} \Gamma_{k,\mu k',\mu'}^{LGN} \sigma_{j,k,\mu} \sigma_{j',k',\mu'} . \qquad (3)$$

Function $V_{jj'}^{VC}$ represents the interaction between synapses at positions $j$ and $j'$ in layer 4 of the primary visual cortex; function $\Gamma_{k,\mu k',\mu'}^{LGN}$ represents the correlation in activity between LGN neurons at positions $k$ and $k'$ of cell types $\mu$ and $\mu'$. The set $B_j$ represents a group of LGN neurons which can project their axons to the position $j$ in the visual cortex; therefore, the magnitude of this set is related to the extent of afferent terminal arborization in the cortex $\lambda^A$.

Taking the above formulation into consideration, we have only to discuss the thermodynamics in the Potts spin system described by the Hamiltonian $H$ at the temperature $T$ in order to discuss the activity-dependent self-organization of afferent neural connections during development.

Next, let us discuss more specific descriptions on the modeling of the visual afferent pathways. We will assume that the LGN serves only as a relay nucleus and that the signal is transferred from the retina to the cortex as if they were directly connected. Therefore, the correlation function $\Gamma_{k,\mu k',\mu'}^{LGN}$ can be treated as that in the retinas $\Gamma_{k,\mu k',\mu'}^{R}$. This function is given by using the lateral interaction function in the retina $V_{k;k'}^{R}$ and the correlation function

of stimuli to RGCs $G_{k_1\,\mu;k_2\,\mu'}$ in the following:

$$\Gamma^R_{k,\mu k',\mu'} = \sum_{k_1,k_2} V^R_{k;k_1} G_{k_1,\mu;\,k_2,\mu'} V^R_{k_2;k'} \ .$$ (4)

For simplicity, the stimuli are treated as white noise:

$$G_{k_1,\mu;\,k_2,\mu'} = \delta_{k_1,k_2} \cdot K_{\mu;\mu'} \ .$$ (5)

Now, we can obtain two models for the formation of afferent synaptic connections between the retinas and the primary visual cortex: model (A), in which ocularity and retinotopy are taken into account:

$$\mu \in \{\text{left, right}\}, \quad \mathbf{K} = \begin{bmatrix} 1 & r_1 \\ r_1 & 1 \end{bmatrix} \ ,$$ (6)

where $r_1$ $(0 \leq r_1 \leq 1)$ is the correlation of activity between the left and right retinas; and model (B), in which on-center and off-center pathways are added to model (A):

$$\mu \in \{(\text{left, on-center}), (\text{left, off-center}), (\text{right, on-center}), (\text{right, off-center})\} \ ,$$

$$\mathbf{K} = \begin{bmatrix} 1 & r_1+r_2 & r_1 & r_1 \\ r_1+r_2 & 1 & r_1 & r_1 \\ r_1 & r_1 & 1 & r_1+r_2 \\ r_1 & r_1 & r_1+r_2 & 1 \end{bmatrix} \ ,$$ (7)

where $r_2$ $(-1 \leq r_2 \leq 1)$ is the correlation of activity between the on-center and off-center RGCs in the same retina when there is no correlation between different retinas. A negative value of $r_2$ means out-of-phase firings between on-center and off-center neurons.

## 3   COMPUTER SIMULATION

Computer simulations were carried out according to the Metropolis algorithm (Metropolis, 1953; Tanaka, 1991). A square panel consisting of 80×80 grids was assumed to be the input layer of the primary visual cortex, where the length of one grid is denoted by $a$. The Potts spin is assigned to each grid. Free boundary conditions were adopted on the border of the panel. One square panel of 20×20 grids was assumed to be a retina for each submodality $\mu$. The length of one grid is given as $4a$ so that the edges for the square model cortex and model retinas are of the same length.

The following form was adopted for the interactions $V^v_{k;k'}$'s $(v = \text{VC or R})$:

$$V^v_{k;k'} = \frac{q^v_{ex}}{2\pi\lambda^v_{ex}{}^2} exp\left(-\frac{d^2_{k,k'}}{2\lambda^v_{ex}{}^2}\right) - \frac{q^v_{inh}}{2\pi\lambda^v_{inh}{}^2} exp\left(-\frac{d^2_{k,k'}}{2\lambda^v_{inh}{}^2}\right) \ .$$ (8)

All results reported in this paper were obtained with parameters whose values are as follows: $q^{VC}_{ex} = 1.0$, $q^{VC}_{inh} = 5.0$, $\lambda^{VC}_{ex} = 0.15$, $\lambda^{VC}_{inh} = 1.0$, $q^{R}_{ex} = 1$, $\lambda^{R}_{ex} = 0.5$, $\lambda^{R}_{inh} = 1.0$, $\lambda^{A} = 1.6$, $a = 0.1$, $T = 0.001$, $r_1 = 0$, and $r_2 = -0.2$. It is assumed that $q^{R}_{inh} = 0$ for model (A) while $q^{R}_{inh} = 0.5$ for model (B). By considering that the receptive field (RF) of an RGC at position $k$ is represented by $\mu V^{R}_{k;k'}$, RGCs for model (A) and (B) have low-pass and high-pass filtering properties, respectively. Monte Carlo simulation for model (A) was carried out for 200,000 steps; that for model (B) was done for 760,000 steps.

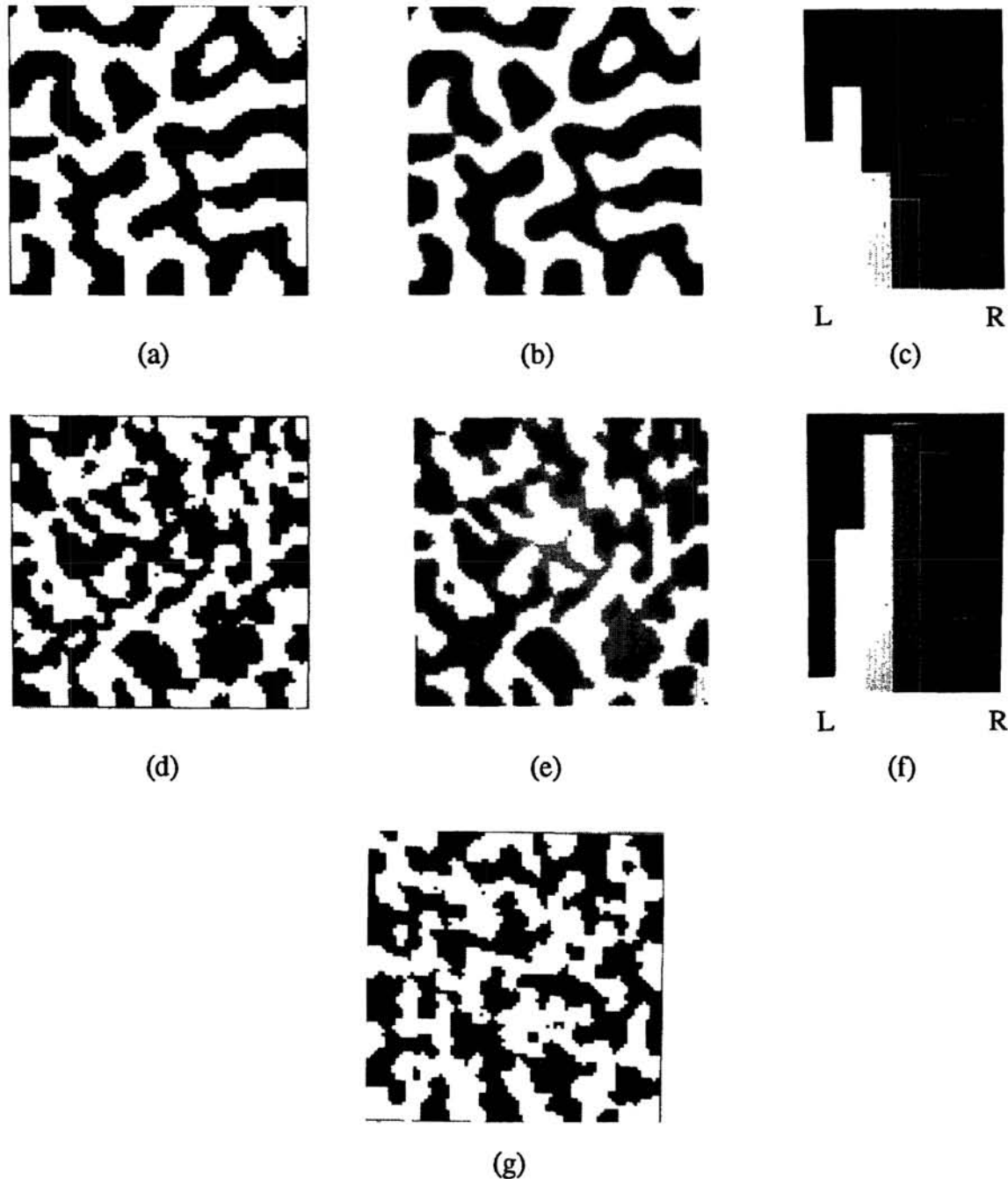

Fig. 1 Simulated results of synaptic terminal and neuronal distributions and ocular dominance histograms for models (A) and (B).

## 4   RESULTS AND DISCUSSIONS

The distributions of synaptic terminals and neurons, and ocular dominance histograms are shown in Fig. 1, where (a), (b) and (c) were obtained from model (A); (d), (e), (f) and (g) were obtained from model (B). The spatial distribution of synaptic terminals originating from the left or right retina (Figs. 1a and 1d) is a counterpart of an autoradiograph of the ODC by the eye-injection of radiolabeled amino acid. The bandwidth of the simulated ODC (Fig. 1a) is almost constant as well as the observed bandwidth for monkeys (Hubel and Wiesel, 1974). The distribution of ocularity in synaptic terminals shown in Fig. 1d is irregular in that the periodicity seen in Fig. 1a disappears even though a patchy pattern can be seen. This pattern is quite similar to the ODC for cats (Anderson, Olavarria and Van Sluyters 1988).

By calculating the convolution of the synaptic connections $\sigma_{j,k,\mu}$'s with the cortical interaction function $V_{j,j'}^{vc}$, the ocular dominance in response of cortical cells to monocular stimulation and the spatial pattern of the ocular dominance in activity (Figs. 1b and 1e) were obtained. Neurons specifically responding to stimuli presented in the right and left eyes are, respectively, in the black and white domains. This pattern is a counterpart of an electrophysiological pattern of the ODC. The distributions of ocularity in synaptic terminals correspond to those of ocular dominance in neuronal response to monocular stimulation (a to b; d to e in Fig. 1). This suggests that the borders of the autoradiographic ODC pattern coincide with those of the electrophysiological ODC pattern. This correspondence is not trivial since strong lateral inhibition exerts in the cortex.

Reflecting the narrow transition areas between monocular domains in Fig. 1b, a dip appears in the binocular bin in the corresponding ODH (Fig. 1c). In contrast, the profile of the ODH (Fig. 1f) has a single peak in the binocular bin since binocularly responsive neurons are distributed over the cortex (Fig. 1e).

In model (B), on-center and off-center terminals are also segregated in the cortex in superposition to the ODC pattern (Fig. 1g). No correlation can be seen between the spatial distribution of on-center/off-center terminals and the ODC pattern (Fig.1d).

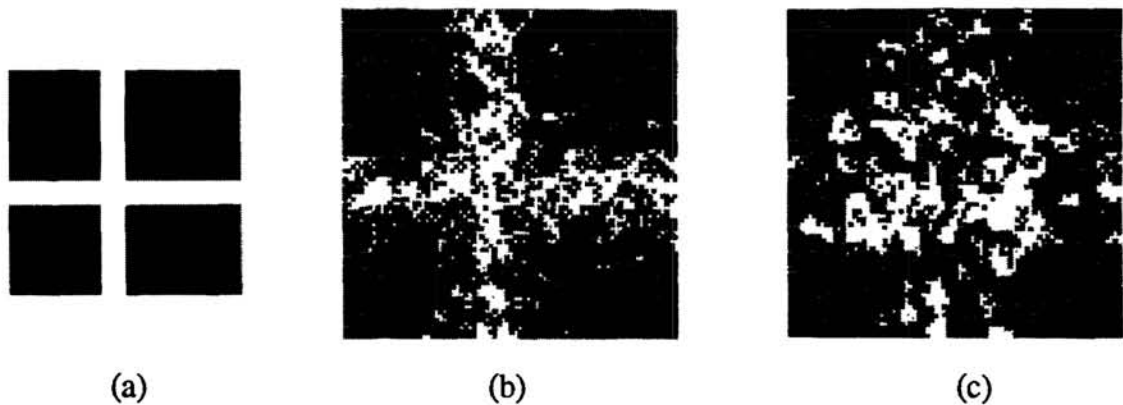

(a)                         (b)                         (c)

Fig. 2 A visual stimulation pattern (a) and the distributions of active synaptic terminals in the cortex [(b) for model (A) and (c) for model (B)].

Figures 2b and 2c visualize spatial patterns of active synaptic terminals in the cortex for model (A) and model (B), when the light stimulus shown by Fig.1d is presented to both

retinas. A pattern similar to the stimulus appears in the cortex for model (A) (Fig. 1e). This supports the observation that retinotopic order is almost achieved. In other simulations for model (A), the retinotopic order in the final pattern was likely to be achieved when initial patterns were roughly ordered in retinotopy. In model (B), the retinotopic order seems to be broken at least in this system size even though the initial pattern has a well-ordered retinotopy (Fig. 1c). There is a tendency for retinotopy to be harder to preserve in model (B) than in model (A).

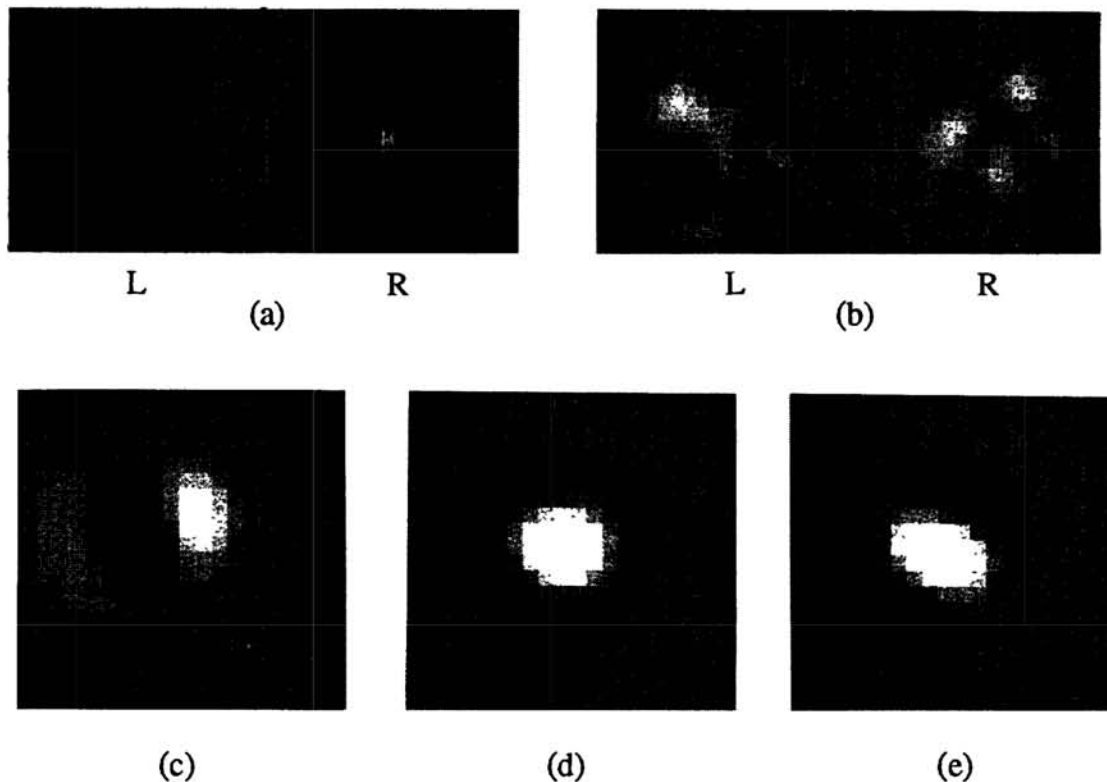

Fig. 3 Representative receptive fields obtained from simulations.

Model (A) reproduced only concentric RFs for both eyes. The dominant RFs of monocular neurons were of the on-center/off-surround type (right in Fig. 3a); the other RFs of the same neurons were of the type of the low-pass filter which has only the off response (left in Fig. 3a). In Model (B), RFs of cortical neurons generally had complex structures (Fig. 3b). It can barely be recognized that the dominant RFs of monocular neurons showed simple-cell-like RFs.

To determine why model (B) produced complex structures in RFs, another simulation of RF formation was carried out based on a model where retinotopy and on-center/off-center pathways are considered. Various types of RFs emerged in the cortex (bottom row in Fig. 3). The difference in structures between Figures. 3c and 3e shows the difference in the orientation and the phase (the deviation of the on region from the RF center) in the simple-cell-like RFs. Fig. 3d shows an on-center concentric RF. Such nonoriented RFs were likely to appear in the vicinity of the singular points around which the orientation rotates by 180 degrees.

Simulations for model (A) with different values of parameters such as $q^{VC}_{inh}$, $\lambda^A$ and $q^R_{inh}$ were also carried out although the results are not visualized here. When $q^{VC}_{inh}$ takes a small

value, the ODC bandwidth fluctuates). However large the fluctuation may be, the left-eye or right-eye dominant domains are well connected, and the pattern does not become an irregular beaded pattern as seen in the cat ODC. When afferent axonal arbors were widely spread in the cortex ($\lambda^A \gg 1$), segregated ODC stripe patterns had only small fluctuation in the bandwidth. $q^R_{inh} = 0$ corresponds to a monotonically decreasing function $V^R_{k:k'}$ with respect to the radial distance $d_{k,k'}$. When $q^R_{inh}$ was increased from zero, the number of monocular neurons was decreased. Therefore, the profile of the ODH changes from that in Fig. 1c.

In model (B), as the value of $r_2$ became smaller, on-center and off-center terminals were more sharply segregated, and the average size of the ODC patches became smaller. The segregation of on-center and off-center terminals seems to interfere strongly with the development of the ODC and the retinotopic organization. This may be attributed to the competition between ocularity and on-center/off-center pathways. We have seen that only concentric or simple-cell-like RFs can be obtained (Fig. 3b) unless both the ocularity and the on-center/off-center pathways are taken into account in simulations. However, in model (B) in which the two types of submodality are treated, neurons have complex separated RF structures (Fig. 3b). This also seems to be due to the competition among the ocularity and the on-center/off-center pathways. The simulation of model (B) was performed with no correlation in activity between the left and right eyes $r_1$. This condition can be realized for binocularly deprived kittens (Tanaka, 1989). By considering this, we may conclude that the formation of normal RFs needs cooperative binocular input.

In this research, we did not consider the effect of color-related cell types on ODC formation. Actually, there are varieties of single-opponent cells in the retina and LGN of monkeys such as four types of red-green opponent cells: a red on-center cell with a green inhibitory surround; a green on-center cell with a red inhibitory surround; a red off-center cell with a green excitatory surround; and a green off-center cell with a red excitatory surround. The correlation of activity between red on-center and green on-center cells or green off-center and red off-center cells may be positive in view of the fact that the spectral response functions between three photoreceptors overlap on the axis of the wavelength. However, the red on-center and green on-center cells antagonize the red off-center and green off-center cells, respectively. Therefore, the former two and latter two can be looked upon as the on-center and off-center cells seen in the retina of cats. This implies that the model for monkeys should be model (B); thereby, the ODC pattern for monkeys should be an irregular beaded pattern despite the fact that the ODC and ODH in model (A) resemble those for monkeys. To avoid such contradiction, the on-center and off-center cells must separately send their axons into different sublayers within layer 4Cβ, as seen in the visual cortex for Tree shrews (Fitzpatrick and Raczkowski, 1990).

## 5  CONCLUSION

In model (A), the ODC showed the striped pattern and the ODH revealed a dip in the binocular bin. In contrast to this, model (B) reproduced spatially modulated irregular ODC patterns and the single-peak behavior of the ODH. From comparison of these simulated results with experimental observations, it is evident that the ODCs and ODHs for models (A) and (B) agree very closely with those seen in monkeys and cats, respectively. Therefore, this leads to the conclusion that model (A) describes the development of the afferent fiber terminals of the primary visual cortex of monkeys, while model (B) describes that of the

cat. In fact, the assumption of the negative correlation ($r_2 < 0$) between the on-center and off-center pathways in model (B) is consistent with the experiments on correlated activity between on-center and off-center RGCs for cats (Mastronarde, 1988).

Finally, we predict the following with regard to afferent projections for cats and monkeys.
[1] In the input layer of the visual cortex for cats, on-center/off-center pathway terminals are segregated into patches, superposing the ocular dominance patterns.
[2] In monkeys, the axons from on-center/off-center cells in the LGN terminate in different sublayers in layer 4C$\beta$ of the primary visual cortex.

## Acknowledgment

The author thanks Mr.Miyashita for his help in performing computer simulations of receptive field formation.

## References

P.A. Anderson, J. Olavarria & R.C. Van Sluyters. (1988) The overall pattern of ocular dominance bands in the cat visual cortex. J. Neurosci., **8**: 2183-2200.

D.H. Hubel, T.N. Wiesel and S. LeVay. (1977). Plasticity of ocular dominance columns in monkey striate cortex. Philos. Trans. R. Soc. Lond., B**278**: 377-409.

T.N. Wiesel and D.H. Hubel. (1974).Ordered arrangement of orientation columns in monkeys lacking visual experience. J. Comp. Neurol. **158**: 307-318

D.H. Hubel and T.N. Wiesel. (1963). Receptive fields, binocular interaction and functional architecture in the cat's visual cortex. J. Physiol., **160**: 106-154.

S. Tanaka. (1990) Theory of self-organization of cortical maps: Mathematical framework. Neural Networks, **3**: 625-640.

N. Metropolis, A. W. Rosenbluth, M. N. Rosenbluth, A. H. Teller and E. Teller. (1953) Equation of state calculations by fast computing machines. J. Chem. Phys., **21**: 1087-1092.

S. Tanaka. (1991) Theory of ocular dominance column formation: Mathematical basis and computer simulation. Biol. Cybern., in press.

S. Tanaka. (1989) Theory of self-organization of cortical maps. In D. S. Touretzky (ed.), Advances in Neural Information Processing Systems 1, 451-458, San Mateo, CA: Morgan Kaufmann.

D. Fitzpatrick and D. Raczkowski. (1990) Innervation patterns of single physiologically identified geniculocortical axons in the striate cortex of the tree shrew. Proc. Natl. Acad. Sci. USA, **87**: 449-453.

D. N. Mastronarde. (1989) Correlated firing of retinal ganglion cells. Trends in Neurosci. **12**: 75-80.